# The Effect of Catecholamines on Performance: From Unit to System Behavior

**David Servan-Schreiber, Harry Printz and Jonathan D. Cohen**
School of Computer Science and Department of Psychology
Carnegie Mellon University
Pittsburgh, PA  15213

## ABSTRACT

At the level of individual neurons, catecholamine release increases the responsivity of cells to excitatory and inhibitory inputs. We present a model of catecholamine effects in a network of neural-like elements. We argue that changes in the responsivity of individual elements do not affect their ability to detect a signal and ignore noise. However, the same changes in cell responsivity in a *network* of such elements do improve the signal detection performance of the network as a whole. We show how this result can be used in a computer simulation of behavior to account for the effect of *CNS* stimulants on the signal detection performance of human subjects.

## 1  Introduction

The catecholamines—norepinephrine and dopamine—are neuroactive substances that are presumed to modulate information processing in the brain, rather than to convey discrete sensory or motor signals. Release of norepinephrine and dopamine occurs over wide areas of the central nervous system, and their post-synaptic effects are long lasting. These effects consist primarily of an enhancement of the response of target cells to other afferent inputs, inhibitory as well as excitatory (see [4] for a review).

Increases or decreases in catecholaminergic tone have many behavioral consequences including effects on motivated behaviors, attention, learning and memory, and motor

behavior. At the information processing level, catecholamines appear to affect the ability to detect a signal when it is imbedded in noise (see review in [3]).

In terms of signal detection theory, this is described as a change in the *performance* of the system. However, there is no adequate account of how these changes at the system level relate to the effect of catecholamines on individual cells. Several investigators [5,12,2] have suggested that catecholamine-mediated increases in a cell's responsivity can be interpreted as a change in the cell's signal-to-noise ratio. By analogy, they proposed that this change at the unit level may account for changes in signal detection performance at the behavioral level.

In the first part of this paper we analyze the relation between unit responsivity, signal-to-noise ratio and signal detection performance in a network of neural elements. We start by showing that the changes in unit responsivity induced by catecholamines do *not* result in changes in signal detection performance of a single unit. We then explain how, in spite of this fact, the aggregate effect of such changes in a *chain* of units can lead to improvements in the signal detection performance of the entire network.

In the second part, we show how changes in gain – which lead to an increase in the signal detection performance of the network – can account for a behavioral phenomenon. We describe a computer simulation of a network performing a signal detection task that has been applied extensively to behavioral research: the continuous performance test. In this simulation, increasing the responsivity of individual units leads to improvements in performance that closely approximate the improvement observed in human subjects under conditions of increased catecholaminergic tone.

## 2   Effect of Gain on a Single Element

We assume that the response of a typical neuron can be described by a strictly increasing function $f_G(x)$ from real-valued inputs to the interval $(0, 1)$. This function relates the strength of a neuron's net afferent input $x$ to its probability of firing, or *activation*. We do not require that $f_G$ is either continuous or differentiable.

For instance, the family of logistics, given by

$$f_G(x) = \frac{1}{1 + e^{-(Gx+B)}}$$

has been proposed as a model of neural activation functions [7,1]. These functions are all strictly increasing, for each value of the *gain* $G > 0$, and all values of the *bias B*.

The potentiating effect of catecholamines on responsivity can be modelled as a change in the shape of its activation function. In the case of the logistic, this is achieved by increasing the value of $G$, as illustrated in Figure 1. However, our analysis applies to *any* suitable family of functions, $\{f_G\}$. We require only that each member function $f_G$ is strictly increasing, and that as $G \to \infty$, the family $\{f_G\}$ converges monotonically to

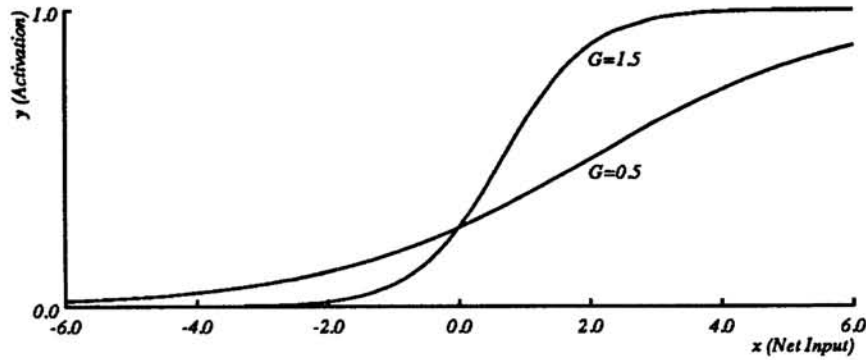

Figure 1: Logistic Activation Function, Used to Model the Response Function of Neurons. Positive net inputs correspond to excitatory stimuli, negative net inputs correspond to inhibitory stimuli. For the graphs drawn here, we set the bias $B$ to $-1$. The asymmetry arising from a negative bias is often found in the response function of actual neurons [6].

the unit step function $u_0$ almost everywhere.[1] Here, $u_0$ is defined as

$$u_0(x) = \begin{cases} 0 & \text{for} \quad x \leq 0 \\ 1 & \text{for} \quad x > 0 \end{cases}$$

This means that as $G$ increases, the value $f_G(x)$ gets steadily closer to 1 if $x > 0$, and steadily closer to 0 if $x \leq 0$.

## 2.1   Gain Does Not Affect Signal Detection Performance

Consider the signal detection performance of a network in which the response of a single unit is compared with a threshold to determine the presence or absence of a signal. We assume that in the presence of the signal, this unit receives a positive (excitatory) net afferent input $x_S$, and in the absence of the signal it receives a null or negative (inhibitory) input $x_A$. When zero-mean noise is added to this quantity, in the presence as well as the absence of the signal, the unit's net input in each case is distributed around $x_S$ or $x_A$ respectively. Therefore its response is distributed around $f_G(x_S)$ or $f_G(x_A)$ respectively (see Figure 2).

In other words, the input in the case where the signal is present is a random variable $X_S$, with probability density function (pdf) $\rho_{X_S}$ and mean $x_S$, and in the absence of the signal it is the random variable $X_A$, with pdf $\rho_{X_A}$ and mean $x_A$. These then determine the random variables $Y_{GS} = f_G(X_S)$ and $Y_{GA} = f_G(X_A)$, with pdfs $\rho_{Y_{GS}}$ and $\rho_{Y_{GA}}$, which represent the *response* in the presence or absence of the signal for a given value of the gain. Figure 2 shows examples of $\rho_{Y_{GS}}$ and $\rho_{Y_{GA}}$ for two different values of $G$, in the case where $f_G$ is the biased logistic.

If the input pdfs $\rho_{X_S}$ and $\rho_{X_A}$ overlap, the output pdfs $\rho_{Y_{GS}}$ and $\rho_{Y_{GA}}$ will also overlap. Thus for any given threshold $\theta$ on the $y$-axis used to categorize the output as "signal present" or "signal absent," there will be some misses and some false alarms. The best

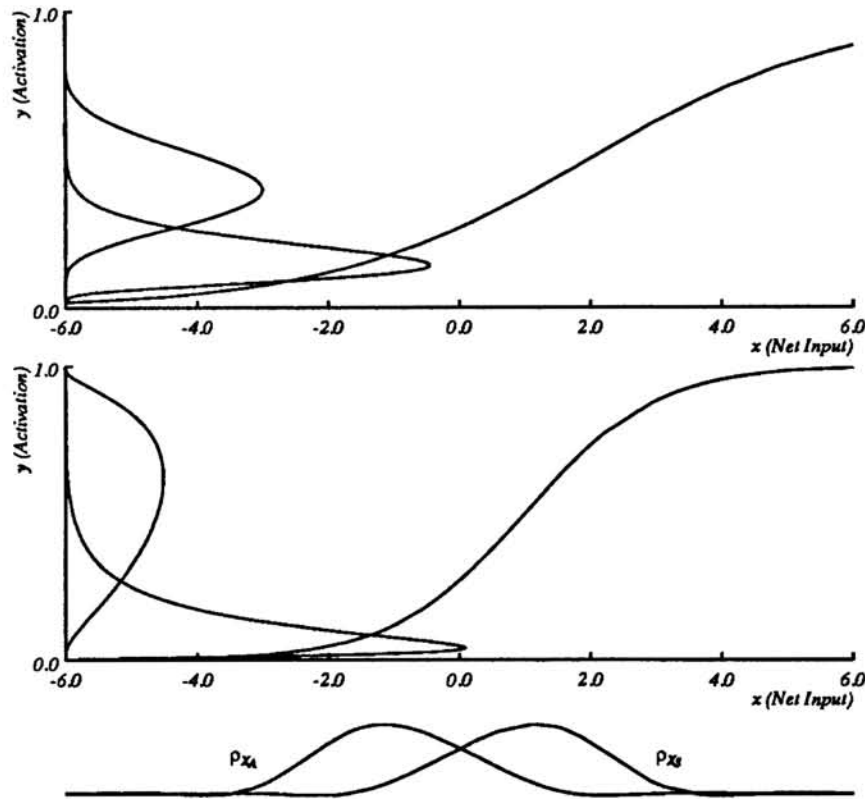

Figure 2: Input and Output Probability Density Functions. The curves at the bottom are the pdfs of the net input in the signal absent (left) and signal present (right) cases. The curves along the y-axis are the response pdfs for each case; they are functions of the activation $y$, and represent the distribution of outputs. The top graph shows the logistic and response pdfs for $G = 0.5$, $B = -1$; the bottom graph shows them for $G = 1.0$, $B = -1$.

the system can do is to select a threshold that optimizes performance. More precisely, the expected payoff or *performance* of the unit is given by

$$E(\theta) = \lambda + \alpha \cdot \Pr(Y_{GS} \geq \theta) - \beta \cdot \Pr(Y_{GA} \geq \theta)$$

where $\lambda$, $\alpha$, and $\beta$ are constants that together reflect the prior probability of the signal, and the payoffs associated with correct detections or *hits*, correct *ignores*, *false alarms* and *misses*. Note that $\Pr(Y_{GS} \geq \theta)$ and $\Pr(Y_{GA} \geq \theta)$ are the probabilities of a hit and a false alarm, respectively.

By solving the equation $dE/d\theta = 0$ we can determine the value $\theta^*$ that maximizes $E$. We call $\theta^*$ the *optimal threshold*. Our first result is that for *any* activation function $f$ that satisfies our assumptions, and *any* fixed input pdfs $\rho_{X_S}$ and $\rho_{X_A}$ the unit's performance at optimal threshold is the same. We call this the Constant Optimal Performance Theorem, which is stated and proved in [10]. In particular, for the logistic, increasing the gain $G$ does *not* induce better performance. It may change the value of the threshold that yields optimal performance, but it does not change the actual performance at optimum. This is because a strictly increasing activation function produces a point-to-point mapping between the distributions of input and output values. Since the amount of overlap between

the two input pdfs $\rho_{X_S}$ and $\rho_{X_A}$ does not change as the gain varies, the amount of overlap in the response pdfs does not change either, even though the shape of the response pdfs does change when gain increases (see Figure 2). [2]

## 3    Effect of Gain on a Chain of Elements

Although increasing the gain does not affect the signal detection performance of a single element, it does improve the performance of a *chain* of such elements. By a chain, we mean an arrangement in which the output of the first unit provides the input to another unit (see Figure 3). Let us call this second element the response unit. We monitor the output of this second unit to determine the presence or absence of a signal.

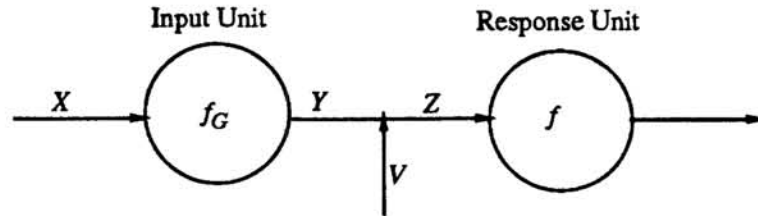

Figure 3: A Chain of Units. The output of the unit receiving the signal is combined with noise to provide input to a second unit, called the *response unit*. The activation of the response unit is compared to a threshold to determine the presence or absence of the signal.

As in the previous discussion, noise is added to the net input to *each* unit in the chain in the presence as well as in the absence of a signal. We represent noise as a random variable $V$, with pdf $\rho_V$ that we assume to be independent of gain. As in the single-unit case, the input to the first unit is a random variable $X_S$, with pdf $\rho_{X_S}$ in the presence of the signal and a random variable $X_A$, with pdf $\rho_{X_A}$ in the absence of the signal. The output of the first unit is described by the random variables $Y_{GS}$ and $Y_{GA}$ with pdfs $\rho_{Y_{GS}}$ and $\rho_{Y_{GA}}$. Now, because noise is added to the net input of the response unit as well, the input of the response unit is the random variable $Z_{GS} = Y_{GS} + V$ or $Z_{GA} = Y_{GA} + V$, again depending on whether the signal is present or absent. We write $\rho_{Z_{GS}}$ and $\rho_{Z_{GA}}$ for the pdfs of these random variables. $\rho_{Z_{GS}}$ is the convolution of $\rho_{Y_{GS}}$ and $\rho_V$, and $\rho_{Z_{GA}}$ is the convolution of $\rho_{Y_{GA}}$ and $\rho_V$. The effect of convolving the output pdfs of the input unit with the noise distribution is to increase the overlap between the resulting distributions ($\rho_{Z_{GS}}$ and $\rho_{Z_{GA}}$), and therefore decrease the discriminability of the input to the response unit.

How are these distributions affected by an increase in gain on the input unit? By the Constant Optimal Performance Theorem, we already know that the overlap between $\rho_{Y_{GS}}$ and $\rho_{Y_{GA}}$ remains constant as gain increases. Furthermore, as stated above, we have assumed that the noise distribution is independent of gain. It would therefore seem that a change in gain should not affect the overlap between $\rho_{Z_{GS}}$ and $\rho_{Z_{GA}}$. However, it is

possible to show that, under very general conditions, the overlap between $\rho_{Z_{GS}}$ and $\rho_{Z_{GA}}$ *decreases* when the gain of the input unit increases, thereby improving performance of the two-layered system. We call this the chain effect; the Chain Performance Theorem [10] gives sufficient conditions for its appearance. [3]

Paradoxically, the chain effect arises because the noise added to the net input of the response unit is not affected by variations in the gain. As we mentioned before, increasing the gain separates the means of the output pdfs of the input unit, $\mu(Y_{GS})$ and $\mu(Y_{GA})$ (eventhough this does not affect the performance of the first unit). Suppose all the probability mass were concentrated at these means. Then $\rho_{Z_{GS}}$ would be a copy of $\rho_V$ centered at $\mu(Y_{GS})$, and $\rho_{Z_{GA}}$ would be a copy of $\rho_V$ centered at $\mu(Y_{GS})$. Thus in this case, increasing the gain *does* correspond to rigidly translating $\rho_{Z_{GS}}$ and $\rho_{Z_{GA}}$ apart, thereby reducing their overlap and improving performance.

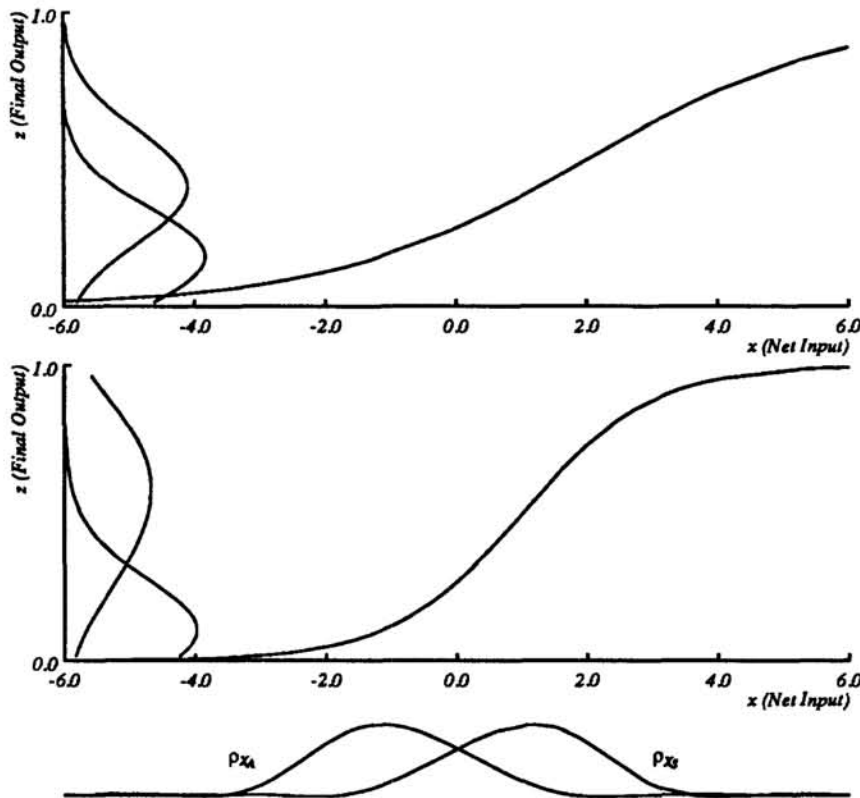

Figure 4: Dependence of Chain Output Pdfs Upon Gain. These graphs use the same conventions and input pdfs as Figure 2. They depict the output pdfs, in the presence of additive Gaussian noise, for $G = 0.5$ (top) and $G = 1.0$ (bottom).

A similar effect arises in more general circumstances, when $\rho_{Y_{GS}}$ and $\rho_{Y_{GA}}$ are not concentrated at their means. Figure 4 provides an example, illustrating $\rho_{Z_{GS}}$ and $\rho_{Z_{GA}}$ for three different values of the gain. The first unit outputs are the same as in Figure 2, but

these have been convolved with the pdf $\rho_V$ of a Gaussian random variable to obtain the curves shown. Careful inspection of the figure will reveal that the overlap between $\rho_{Z_{GS}}$ and $\rho_{Z_{GA}}$ decreases as the gain rises.

## 4   Simulation of the Continuous Performance Test

The above analysis has shown that increasing the gain of the response function of individual units in a very simple network can improve signal detection performance. We now present computer simulation results showing that this phenomenon may account for improvements of performance with catecholamine agonists in a common behavioral test of signal detection.

The continous performance test (CPT) has been used extensively to study attention and vigilance in behavioral and clinical research. Performance on this task has been shown to be sensitive to drugs or pathological conditions affecting catecholamine systems [11,8,9]. In this task, individual letters are displayed tachystoscopically in a sequence on a computer monitor. In one common version of the task, a target event is to be reported when two consecutive letters are identical. During baseline performance, subjects typically fail to report 10 to 20% of targets ("misses") and inappropriately report a target during 0.5 to 1% of the remaining events ("false alarms"). Following the administration of agents that directly release catecholamines from synaptic terminals and block re-uptake from the synaptic cleft (i.e., CNS stimulants such as amphetamines or methylphenidate) the number of misses decreases, while the number of false alarms remains approximately the same. Using standard signal detection theory measures, investigators have claimed that this pattern of results reflects an improvement in the discrimination between signal and non-signal events ($d'$), while the response criterion ($\beta$) does not vary significantly [11,8,9].

We used the backpropagation learning algorithm to train a recurrent three layer network to perform the CPT (see Figure 5). In this model, several simplifyng assumptions made in the preceding section are removed: in contrast to the simple two-unit assembly, the network contains three layers of units (input layer, intermediate – or hidden – layer, and output layer) with some recurrent connections; connection weights between these layers are developed entirely by the training procedure; as a result, the activation patterns on the intermediate layer that are evoked by the presence or absence of a signal are also determined solely by the training procedure; finally, the representation of the signal is distributed over an ensemble of units rather than determined by a single unit.

Following training, Gaussian noise with zero mean was added to the net input of each unit in the intermediate and output layers as each letter was presented. The overall standard deviation of the noise distribution and the threshold of the response unit were adjusted to produce a performance equivalent to that of subjects under baseline conditions (13.0% misses and 0.75% false alarms). We then increased the gain of all the intermediate and output units from 1.0 to 1.1 to simulate the effect of catecholamine release in the network. This manipulation resulted in rates of 6.6% misses and 0.78% false alarms. The correspondence between the network's behavior and empirical data is illustrated in Figure 5.

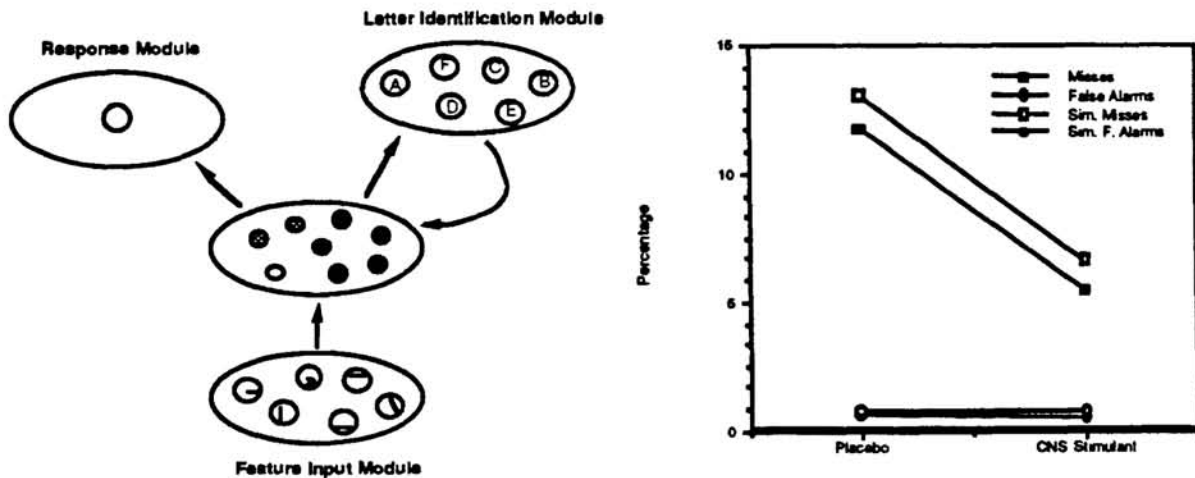

Figure 5: Simulation of the Continuous Performance Task. **Left panel:** The recurrent three-layer network (12 input units, 30 intermediate units, 10 output units and 1 response unit). Each unit projects to all units in the subsequent layer. In addition, each output unit also projects to each unit in the intermediate layer. The gain parameter $G$ is the same for all intermediate and output units. In the simulation of the placebo condition, $G = 1$; in the simulation of the drug condition, $G = 1.1$. The bias $B = -1$ in both conditions. **Right panel:** Performance of human subjects [9], and of the simulation, on the CPT. With methylphenidate misses dropped from 11.7% to 5.5%, false alarms decreased from 0.6% to 0.5% (non-significant).

The enhancement of signal detection performance in the simulation is a robust effect. It appears when gain is increased in the intermediate layer only, in the output layer only, or in both layers. Because of the recurrent connections between the output layer and the intermediate layer, a chain effect occurs between these two layers when the gain is increased over any one of them, or both of them. The impact of the chain effect is to reduce the distortion, due to internal noise, of the distributed representation on the layer receiving inputs from the layer where gain is increased. Note also that the improvement takes place even though there is no noise added to the input of the response unit. The response unit in this network acts only as an indicator of the strength of the signal in the intermediate layer. Finally, as the Constant Optimal Performance Theorem predicts, increasing the gain only on the response unit does not affect the performance of the network.

## 5   Conclusion

Fluctuations in catecholaminergic tone accompany psychological states such as arousal, motivation and stress. Furthermore, dysfunctions of catecholamine systems are implicated in several of the major psychiatric disorders. However, in the absence of models relating changes in cell function to changes in system performance, the relation of catecholamines to behavior has remained obscure. The findings reported in this paper suggest that the behavioral impact of catecholamines depend on their effects on an ensemble of units operating in the presence of noise, and not just on changes in individual unit responses.

Furthermore, they indicate how neuromodulatory effects can be incorporated in parallel distributed processing models of behavior.

## Footnotes

[1]A sequence of functions $\{g_n\}$ converges almost everywhere to the function $g$ if the set of points where it diverges, or converges to the wrong value, is of measure zero.

[2]We present the intuitions underlying our results in terms of the overlap between the pdfs. However, the proofs themselves are analytical.

[3]In this discussion, we have assumed that the same noise was added to the net input into each unit of a chain. However, the improvement in performance of a chain of units with increasing gain does not depend on this particular assumption.

## References

[1] Y. Burnod and H. Korn. Consequences of stochastic release of neurotransmitters for network computation in the central nervous system. *Proceedings of the National Academy of Science*, 86:352–356, 1988.

[2] L. A. Chiodo and T. W. Berger. Interactions between dopamine and amino acid-induced excitation and inhibition in the striatum. *Brain Research*, 375:198–203, 1986.

[3] C. R. Clark, G. M. Geffen, and L. B. Geffen. Catecholamines and attention ii: pharmacological studies in normal humans. *Neuroscience and Behavioral Reviews*, 11:353–364, 1987.

[4] S. L. Foote. Extrathalamic modulation of cortical function. *Ann. Rev. Neurosci.*, 10:67–95, 1987.

[5] S. L. Foote, R. Freedman, and A. P. Olivier. Effects of putative neurotransmitters on neuronal activity in monkey auditory cortex. *Brain Research*, 86:229–242, 1975.

[6] W. J. Freeman. Nonlinear gain mediating cortical stimulus-response relations. *Biological Cybernetics*, 33:243–247, 1979.

[7] G. E Hinton and Sejnowski T. J. Analyzing cooperative computation. *Proceedings of the Cognitive Science Society*, 1983.

[8] R. Klorman, L. O. Bauer, H. W. Coons, J. L. Lewis, L. J. Peloquin, R. A. Perlmutter, R. M. Ryan, L. F. Salzman, and J. Strauss. Enhancing effects of methylphenidate on normal young adults cognitive processes. *Psychopharmacology Bulletin*, 20:3–9, 1984.

[9] L. J. Peloquin and R. Klorman. Effects of methylphenidate on normal children's mood, event-related potentials, and performance in memory scanning and vigilance. *Journal of Abnormal Psychology*, 95:88–98, 1986.

[10] H. Printz and D. Servan-Schreiber. *Foundations of a Computational Theory of Catecholamine Effects*. Technical Report CMU-CS-90-105, Carnegie Mellon, School of Computer Science, 1990.

[11] J. Rapoport, M. S. Buchsbaum, H. Weingartner, T. P. Zahn, C. Ludlow, J. Bartko, E. J. Mikkelsen, D. H. Langer, and Bunney W. E. Dextroamphetamine: cognitive and behavioral effects in normal and hyperactive boys and normal adult males. *Archives of General Psychiatry*, 37:933–943, 1980.

[12] M. Segal. Mechanisms of action of noradrenaline in the brain. *Physiological Psychology*, 13:172–178, 1985.